# Predicting Lifetimes in Dynamically Allocated Memory

**David A. Cohn**
Adaptive Systems Group
Harlequin, Inc.
Menlo Park, CA 94025
cohn@harlequin.com

**Satinder Singh**
Department of Computer Science
University of Colorado
Boulder, CO 80309
baveja@cs.colorado.edu

## Abstract

Predictions of lifetimes of dynamically allocated objects can be used to improve time and space efficiency of dynamic memory management in computer programs. Barrett and Zorn [1993] used a simple lifetime predictor and demonstrated this improvement on a variety of computer programs. In this paper, we use decision trees to do lifetime prediction on the same programs and show significantly better prediction. Our method also has the advantage that during training we can use a large number of features and let the decision tree automatically choose the relevant subset.

## 1 INTELLIGENT MEMORY ALLOCATION

Dynamic memory allocation is used in many computer applications. The application requests blocks of memory from the operating system or from a memory manager when needed and explicitly frees them up after use. Typically, all of these requests are handled in the same way, without any regard for how or for how long the requested block will be used. Sometimes programmers use runtime profiles to analyze the typical behavior of their program and write special purpose memory management routines specifically tuned to dominant classes of allocation events. Machine learning methods offer the opportunity to automate the process of tuning memory management systems.

In a recent study, Barrett and Zorn [1993] used two allocators: a special allocator for objects that are short-lived, and a default allocator for everything else. They tried a simple prediction method on a number of public-domain, allocation-intensive programs and got mixed results on the lifetime prediction problem. Nevertheless, they showed that for all the cases where they were able to predict well, their strategy of assigning objects predicted to be short-lived to the special allocator led to savings

in program running times. Their results imply that if we could predict well in all cases we could get similar savings for all programs. We concentrate on the lifetime prediction task in this paper and show that using axis-parallel decision trees does indeed lead to significantly better prediction on all the programs studied by Zorn and Grunwald and some others that we included. Another advantage of our approach is that we can use a large number of features about the allocation requests and let the decision tree decide on their relevance.

There are a number of advantages of using lifetime predictions for intelligent memory management. It can improve CPU usage, by using special-purpose allocators, e.g., short-lived objects can be allocated in small spaces by incrementing a pointer and deallocated together when they are all dead. It can decrease memory fragmentation, because the short-lived objects do not pollute the address space of long lived objects. Finally, it can improve program locality, and thus program speed, because the short-lived objects are all allocated in a small part of the heap.

The advantages of prediction must be weighed against the time required to examine each request and make that prediction about its intended use. It is frequently argued that, as computers and memory become faster and cheaper, we need to be less concerned about the speed and efficiency of machine learning algorithms. When the purpose of the algorithm is to save space and computation, however, these concerns are paramount.

## 1.1  RELATED WORK

Traditionally, memory management has been relegated to a single, general-purpose allocator. When performance is critical, software developers will frequently build a custom memory manager which they believe is tuned to optimize the performance of the program. Not only is this hand construction inefficient in terms of the programming time required, this "optimization" may seriously degrade the program's performance if it does not accurately reflect the program's use [Wilson et al., 1995].

Customalloc [Grunwald and Zorn, 1992] monitors program runs on benchmark inputs to determine the most commonly requested block sizes. It then produces a set of memory allocation routines which are customized to efficiently allocate those block sizes. Other memory requests are still handled by a general purpose allocator.

Barrett and Zorn [1993] studied lifetime prediction based on benchmark inputs. At each allocation request, the call graph (the list of nested procedure/function calls in effect at the time) and the object size was used to identify an *allocation site*. If all allocations from a particular site were short-lived on the benchmark inputs, their algorithm predicted that future allocations would also be short-lived. Their method produced mixed results at lifetime prediction, but demonstrated the savings that such predictions could bring.

In this paper, we discuss an approach to lifetime prediction which uses learned decision trees. In the next section, we first discuss the identification of relevant state features by a decision tree. Section 3 discusses in greater detail the problem of lifetime prediction. Section 4 describes the empirical results of applying this approach to several benchmark programs, and Section 5 discusses the implications of these results and directions for future work.

## 2  FEATURE SELECTION WITH A DECISION TREE

Barrett and Zorn's approach captures state information in the form of the program's call graph at the time of an allocation request, which is recorded to a fixed predetermined depth. This graph, plus the request size, specifies an allocation "site"; statistics are gathered separately for each site. A drawback of this approach is that it forces a division for each distinct call graph, preventing generalization across irrelevant features. Computationally, it requires maintaining an explicit call graph (information that the program would not normally provide), as well as storing a potentially large table of call sites from which to make predictions. It also ignores other potentially useful information, such as the *parameters* of the functions on the call stack, and the contents of heap memory and the program registers at the time of the request.

Ideally, we would like to examine as much of the program state as possible at the time of each allocation request, and automatically extract those pieces of information that best allow predicting how the requested block will be used. Decision tree algorithms are useful for this sort of task. A decision tree divides inputs on basis of how each input feature improves "purity" of the tree's leaves. Inputs that are statistically irrelevant for prediction are not used in any splits; the tree's final set of decisions examine only input features that improve its predictive performance.

Regardless of the parsimony of the final tree however, training a tree with the entire program state as a feature vector is computationally infeasible. In our experiments, detailed below, we arbitrarily used the top 20 words on the stack, along with the request size, as an approximate indicator of program state. On the target machine (a Sparcstation), we found that including program registers in the feature set made no significant difference, and so dropped them from consideration for efficiency.

## 3  LIFETIME PREDICTION

The characteristic of memory requests that we would like to predict is the lifetime of the block – how long it will be before the requested memory is returned to the central pool. Accurate lifetime prediction lets one segregate memory into short-term, long-term and permanent storage. To this end, we have used a decision tree learning algorithm to derive rules that distinguish "short-lived" and "permanent" allocations from the general pool of allocation requests.

For short-lived blocks, one can create a very simple and efficient allocation scheme [Barrett and Zorn, 1993]. For "permanent" blocks, allocation is also simple and cheap, because the allocator does not need to compute and store any of the information that would normally be required to keep track of the block and return it to the pool when freed.

One complication is that of unequal loss for different types of incorrect predictions. An appropriately routed memory request may save dozens of instruction cycles, but an inappropriately routed one may cost hundreds. The cost in terms of memory may also be unequal: a short-lived block that is incorrectly predicted to be "permanent" will permanently tie up the space occupied by the block (if it is allocated via a method that can not be freed). A "permanent" block, however, that is incorrectly predicted to be short-lived may pollute the allocator's short-term space by preventing a large segment of otherwise free memory from being reclaimed (see Barrett and Zorn for examples).

These risks translate into a time-space tradeoff that depends on the properties of

the specific allocators used and the space limitations of the target machine. For our experiments, we arbitrarily defined false positives and false negatives to have equal loss, except where noted otherwise. Other cases may be handled by reweighting the splitting criterion, or by rebalancing the training inputs (as described in the following section).

# 4   EXPERIMENTS

We conducted two types of experiments. The first measured the ability of learned decision trees to predict allocation lifetimes. The second incorporated these learned trees into the target applications and measured the change in runtime performance.

## 4.1   PREDICTIVE ACCURACY

We used the OC1 decision tree software (designed by Murthy et al. [1994]) and considered only axis-parallel splits, in effect, conditioning each decision on a single stack feature. We chose the sum minority criterion for splits, which minimizes the number of training examples misclassified after the split. For tree pruning, we used the cost complexity heuristic, which holds back a fraction (in our case 10%) of the data set for testing, and selects the smallest pruning of the original tree that is within one standard error squared of the best tree [Breiman et al. 1984]. The details of these and other criteria may be found in Murthy et al. [1994] and Breiman et al. [1984]. In addition to the automatically-pruned trees, we also examined trees that had been truncated to four leaves, in effect examining no more than two features before making a decision.

OC1 includes no provisions for explicitly specifying a loss function for false positive and false negative classifications. It would be straightforward to incorporate this into the sum minority splitting criterion; we chose instead to incorporate the loss function into the training set itself, by duplicating training examples to match the target ratios (in our case, forcing an equal number of positive and negative examples).

In our experiments, we used the set of benchmark applications reported on by Barrett and Zorn: *Ghostscript*, a PostScript interpreter, *Espresso*, a PLA logic optimizer, and *Cfrac*, a program for factoring large numbers, *Gawk*, an AWK programming language interpreter and *Perl*, a report extraction language. We also examined *Gcc*, a public-domain C compiler, based on our company's specific interest in compiler technology.

The experimental procedure was as follows: We linked the application program with a modified *malloc* routine which, in addition to allocating the requested memory, wrote to a trace file the size of the requested block, and the top 20 machine words on the program stack. Calls to *free* allowed tagging the existing allocations, which, following Barrett and Zorn, were labeled according to how many bytes had been allocated during their lifetime.[1]

It is worth noting that these experiments were run on a Sparcstation, which frequently optimizes away the traditional stack frame. While it would have been possible to force the system to maintain a traditional stack, we wished to work from whatever information was available from the program "in the wild", without overriding system optimizations.

Input files were taken from the public ftp archive made available by Zorn and Grunwald [1993]. Our procedure was to take traces of three of the files (typically the largest three for which we could store an entire program trace). Two of the traces were combined to form a training set for the decision tree, and the third was used to test the learned tree.

**Ghostscript** training files: manual.ps and large.ps; test file: ud-doc.ps

**Espresso** training files: cps and mlp4; test file: Z5xp1

**Cfrac** training inputs: 41757646344123832613190542166099121 and 32790560674042145883190; test input: 41757646344124860145938030287

**Gawk** training file: adj.awk/words-small.awk; test file: adj.awk/words-large.awk[2]

**Perl** training files: endsort.perl (endsort.perl as input), hosts.perl (hosts-data.perl as input); test file: adj.perl(words-small.awk as input)

**Gcc** training files: cse.c and combine.c; test file: expr.c

### 4.1.1  SHORT-LIVED ALLOCATIONS

First, we attempted to distinguish short-lived allocations from the general pool. For comparison with Barrett and Zorn [1993], we defined "short-lived" allocations as those that were freed before 32k subsequent bytes had been allocated. The experimental results of this section are summarized in Table 1.

| application | Barrett & Zorn | | OC1 | | | |
|---|---|---|---|---|---|---|
| | false pos % | false neg % | false pos % | | false neg % | |
| ghostscript | 0 | 25.2 | 0.13 | (0.72) | 1.7 | (13.5) |
| espresso | 0.006 | 72 | 0.38 | (1.39) | 6.58 | (14.9) |
| cfrac | 3.65 | 52.7 | 2.5 | (0.49) | 16.9 | (19.4) |
| gawk | 0 | -[3] | 0.092 | (0.092) | 0.34 | (0.34) |
| perl | 1.11 | 78.6 | 5.32 | (10.8) | 33.8 | (34.3) |
| gcc | - | - | 0.85 | (2.54) | 31.1 | (31.0) |

Table 1: Prediction errors for "short-lived" allocations, in percentages of misallocated bytes. Values in parentheses are for trees that have been truncated to two levels. Barrett and Zorn's results included for comparison where available.

### 4.1.2  "PERMANENT" ALLOCATIONS

We then attempted to distinguish "permanent" allocations from the general pool (Barrett and Zorn only consider the short-lived allocations discussed in the previous section). "Permanent" allocations were those that were not freed until the program terminated. Note that there is some ambiguity in these definitions — a "permanent" block that is allocated near the end of the program's lifetime may also be "short-lived". Table 2 summarizes the results of these experiments.

We have not had the opportunity to examine the function of each of the "relevant features" in the program stacks; this is a subject for future work.

| application | false pos % | false neg % |
|---|---|---|
| ghostscript | 0 | 0.067 |
| espresso | 0 | 1.27 |
| cfrac | 0.019 | 3.3 |
| gcc | 0.35 | 19.5 |

Table 2: Prediction errors for "permanent" allocations (% misallocated bytes).

## 4.2  RUNTIME PERFORMANCE

The raw error rates we have presented above indicate that it is possible to make accurate predictions about the lifetime of allocation requests, but not whether those predictions are good enough to improve program performance. To address that question, we have incorporated predictive trees into three of the above applications and measured the effect on their runtimes.

We used a hybrid implementation, replacing the single monolithic decision tree with a number of simpler, site-specific trees. A "site" in this case was a lexical instance of a call to malloc or its equivalent. When allocations from a site were exclusively short-lived or permanent, we could directly insert a call to one of the specialized allocators (in the manner of Barrett and Zorn). When allocations from a site were mixed, a site-specific tree was put in place to predict the allocation lifetime.

Requests predicted to be short-lived were routed to a "quick malloc" routine similar to the one described by Barrett and Zorn; those predicted to be permanent were routed to another routine specialized for the purpose. On tests with random data these specialized routines were approximately four times faster than "malloc".

Our experiments targeted three applications with varying degrees of predictive accuracy: ghostscript, gcc, and cfrac. The results are encouraging (see Table 3). For ghostscript and gcc, which have the best predictive accuracies on the benchmark data (from Section 4.1), we had a clear improvement in performance. For cfrac, with much lower accuracy, we had mixed results: for shorter runs, the runtime performance was improved, but on longer runs there were enough missed predictions to pollute the short-lived memory area and degrade performance.

## 5  DISCUSSION

The application of machine learning to computer software and operating systems is a largely untapped field with promises of great benefit. In this paper we have described one such application, producing efficient and accurate predictions of the lifetimes of memory allocations.

Our data suggest that, even with a feature set as large as a runtime program stack, it is possible to characterize and predict the memory usage of a program after only a few benchmark runs. The exceptions appear to be programs like Perl and gawk which take both a script and a data file. Their memory usage depends not only upon characterizing typical scripts, but the typical data sets those scripts act upon.[4]

Our ongoing research in memory management is pursuing a number of other con-

| application | benchmark test error | | | run time | |
|---|---|---|---|---|---|
| (training set) | short | long | permanent | normal | predictive |
| ghostscript, trained on ud-doc.ps; 7 sites, 1 tree | | | | | |
| manual.ps | 16/256432 | 0/3431 | 0/0 | 96.29 | 95.43 |
| large.ps | | | | 17.22 | 16.75 |
| thesis.ps | | | | 40.27 | 37.57 |
| gcc, trained on combine, cse, c-decl; 17 sites, 4 trees | | | | | |
| expr.c | 0/11988 | 2786/11998 | 301/536875 | 12.59 | 12.40 |
| loop.c | | | | 5.16 | 5.16 |
| reload1.c | | | | 7.02 | 6.81 |
| cfrac, trained on 100···057; 8 sites, 4 trees | | | | | |
| 327···903 | 24/7970099 | 13172/22332 | 106/271 | 7.75 | 7.23 |
| 417···771 | | | | 67.93 | 74.57 |
| 417···121 | | | | 225.31 | 245.64 |

Table 3: Running times in seconds for applications with site-specific trees. Times shown are averages over 24-40 runs, and with the exception of loop.c, are statistically significant with probability greater than 99%.

tinuations of the results described here, including lifetime clustering and intelligent garbage collection.

## Footnotes

[1]We have also examined, with comparable success, predicting lifetimes in terms of the number of intervening calls to malloc, which may be argued as an equally useful measure. We focus on number of bytes for the purposes of comparison with the existing literature.

[2]For *Gawk*, we varied the training to match that used by Barrett and Zorn. They used as training input a single gawk program file run with one data set, and tested on the same gawk program run with another.

[3]We were unable to compute Barrett and Zorn's exact results here, although it appears that their false negative rate was less than 1%.

[4]Perl's generalization performance is significantly better when tested on the same script with different data. We have reported the results using different scripts for comparison with Barrett and Zorn.

## REFERENCES

**D. Barrett and B. Zorn** (1993) Using lifetime predictors to improve memory allocation performance. SIGPLAN'93 – Conference on Programming Language Design and Implementation, June 1993, Albuquerque, New Mexico, pp. 187-196.

**L. Breiman, J. Friedman, R. Olshen and C. Stone** (1984) *Classification and Regression Trees*, Wadsworth International Group, Belmont, CA.

**D. Grunwald and B. Zorn** (1992) CUSTOMALLOC: Efficient synthesized memory allocators. Technical Report CU-CS-602-92, Dept. of Computer Science, University of Colorado.

**S. Murthy, S. Kasif and S. Salzberg** (1994) A system for induction of oblique decision trees. *Journal of Artificial Intelligence Research* 2:1–32.

**P. Wilson, M. Johnstone, M. Neely and D. Boles** (1995) Dynamic storage allocation: a survey and critical review. Proc. 1995 Intn'l Workshop on Memory Management, Kinross, Scotland, Sept. 27–29, Springer Verlag.

**B. Zorn and D. Grunwald** (1993) A set of benchmark inputs made publicly available, in ftp archive `ftp.cs.colorado.edu:/pub/misc/malloc-benchmarks/`.